# Intrinsically Motivated Reinforcement Learning

**Satinder Singh**
Computer Science & Eng.
University of Michigan
baveja@umich.edu

**Andrew G. Barto**
Dept. of Computer Science
University of Massachusetts
barto@cs.umass.edu

**Nuttapong Chentanez**
Computer Science & Eng.
University of Michigan
nchentan@umich.edu

## Abstract

Psychologists call behavior intrinsically motivated when it is engaged in for its own sake rather than as a step toward solving a specific problem of clear practical value. But what we learn during intrinsically motivated behavior is essential for our development as competent autonomous entities able to efficiently solve a wide range of practical problems as they arise. In this paper we present initial results from a computational study of *intrinsically motivated reinforcement learning* aimed at allowing artificial agents to construct and extend hierarchies of reusable skills that are needed for competent autonomy.

## 1 Introduction

Psychologists distinguish between *extrinsic motivation*, which means being moved to do something because of some specific rewarding outcome, and *intrinsic motivation*, which refers to being moved to do something because it is inherently enjoyable. Intrinsic motivation leads organisms to engage in exploration, play, and other behavior driven by curiosity in the absence of explicit reward. These activities favor the development of broad competence rather than being directed to more externally-directed goals (e.g., ref. [14]). In contrast, machine learning algorithms are typically applied to single problems and so do not cope flexibly with new problems as they arise over extended periods of time.

Although the acquisition of competence may not be driven by specific problems, this competence is routinely enlisted to solve many different specific problems over the agent's lifetime. The skills making up general competence act as the "building blocks" out of which an agent can form solutions to new problems as they arise. Instead of facing each new challenge by trying to create a solution out of low-level primitives, it can focus on combining and adjusting its higher-level skills. In animals, this greatly increases the efficiency of learning to solve new problems, and our main objective is to achieve a similar efficiency in our machine learning algorithms and architectures.

This paper presents an elaboration of the reinforcement learning (RL) framework [11] that encompasses the autonomous development of skill hierarchies through *intrinsically motivated reinforcement learning*. We illustrate its ability to allow an agent to learn broad competence in a simple "playroom" environment. In a related paper [1], we provide more extensive background for this approach, whereas here the focus is more on algorithmic details.

Lack of space prevents us from providing a comprehensive background to the many ideas to which our approach is connected. Many researchers have argued for this kind of devel-

opmental approach in which an agent undergoes an extended developmental period during which collections of reusable skills are autonomously learned that will be useful for a wide range of later challenges (e.g., [4, 13]). The previous machine learning research most closely related is that of Schmidhuber (e.g., [8]) on confidence-based curiosity and the ideas of exploration and shaping bonuses [6, 10], although our definition of intrinsic reward differs from these. The most direct inspiration behind the experiment reported in this paper, comes from neuroscience. The neuromodulator dopamine has long been associated with reward learning [9]. Recent studies [2, 3] have focused on the idea that dopamine not only plays a critical role in the extrinsic motivational control of behaviors aimed at harvesting explicit rewards, but also in the intrinsic motivational control of behaviors associated with novelty and exploration. For instance, salient, novel sensory stimuli inspire the same sort of phasic activity of dopamine cells as unpredicted rewards. However, this activation extinguishes more or less quickly as the stimuli become familiar. This may underlie the fact that novelty itself has rewarding characteristics [7]. These connections are key components of our approach to intrinsically motivated RL.

## 2   Reinforcement Learning of Skills

According to the "standard" view of RL (e.g., [11]) the agent-environment interaction is envisioned as the interaction between a controller (the agent) and the controlled system (the environment), with a specialized reward signal coming from a "critic" in the environment that evaluates (usually with a scalar reward value) the agent's behavior (Fig. 1A). The agent learns to improve its skill in controlling the environment in the sense of learning how to increase the total amount of reward it receives over time from the critic.

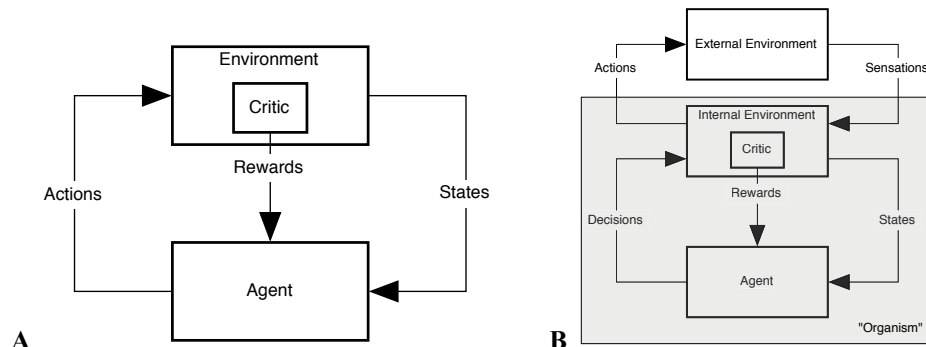

Figure 1: *Agent-Environment Interaction in RL.* **A***: The usual view.* **B***: An elaboration.*

Sutton and Barto [11] point out that one should not identify this RL agent with an entire animal or robot. An an animal's reward signals are determined by processes within its brain that monitor not only external state but also the animal's internal state. The critic is in an animal's head. Fig. 1B makes this more explicit by "factoring" the environment of Fig. 1A into an *external environment* and an *internal environment*, the latter of which contains the critic which determines primary reward. This scheme still includes cases in which reward is essentially an external stimulus (e.g., a pat on the head or a word of praise). These are simply stimuli transduced by the internal environment so as to generate the appropriate level of primary reward.

The usual practice in applying RL algorithms is to formulate the problem one wants the agent to learn how to solve (e.g., win at backgammon) and define a reward function specially tailored for this problem (e.g., reward = 1 on a win, reward = 0 on a loss). Sometimes considerable ingenuity is required to craft an appropriate reward function. The point of departure for our approach is to note that the internal environment contains, among other things, the organism's motivational system, *which needs to be a sophisticated system that*

*should not have to be redesigned for different problems*. Handcrafting a different special-purpose motivational system (as in the usual RL practice) should be largely unnecessary.

**Skills**—Autonomous mental development should result in a collection of reusable skills. But what do we mean by a skill? Our approach to skills builds on the theory of *options* [12]. Briefly, an option is something like a subroutine. It consists of 1) an *option policy* that directs the agent's behavior for a subset of the environment states, 2) an *initiation set* consisting of all the states in which the option can be initiated, and 3) a *termination condition*, which specifies the conditions under which the option terminates. It is important to note that an option is not a sequence of actions; it is a closed-loop control rule, meaning that it is responsive to on-going state changes. Furthermore, because options can invoke other options as actions, hierarchical skills and algorithms for learning them naturally emerge from the conception of skills as options. Theoretically, when options are added to the set of admissible agent actions, the usual Markov decision process (MDP) formulation of RL extends to semi-Markov decision processes (SMDPs), with the one-step actions now becoming the "primitive actions." All of the theory and algorithms applicable to SMDPs can be appropriated for decision making and learning with options [12].

Two components of the the options framework are especially important for our approach:
*1. Option Models*: An option model is a probabilistic description of the effects of executing an option. As a function of an environment state where the option is initiated, it gives the probability with which the option will terminate at any other state, and it gives the total amount of reward expected over the option's execution. Option models can be learned from experience (usually only approximately) using standard methods. Option models allow stochastic planning methods to be extended to handle planning at higher levels of abstraction.
*2. Intra-option Learning Methods*: These methods allow the policies of many options to be updated simultaneously during an agent's interaction with the environment. If an option *could have* produced a primitive action in a given state, its policy can be updated on the basis of the observed consequences even though it was not directing the agent's behavior at the time.

*In most of the work with options, the set of options must be provided by the system designer.* While an option's policy can be improved through learning, each option has to be predefined by providing its initiation set, termination condition, and the reward function that evaluates its performance. Many researchers have recognized the desirability of automatically creating options, and several approaches have recently been proposed (e.g., [5]). For the most part, these methods extract options from the learning system's attempts to solve a particular problem, whereas our approach creates options outside of the context of solving any particular problem.

**Developing Hierarchical Collections of Skills**—Children accumulate skills while they engage in intrinsically motivated behavior, e.g., while at play. When they notice that something they can do reliably results in an interesting consequence, they remember this in a form that will allow them to bring this consequence about if they wish to do so at a future time when they think it might contribute to a specific goal. Moreover, they improve the efficiency with which they bring about this interesting consequence with repetition, before they become bored and move on to something else. *We claim that the concepts of an option and an option model are exactly appropriate to model this type of behavior.* Indeed, one of our main contributions is a (preliminary) demonstration of this claim.

## 3   Intrinsically Motivated RL

Our main departure from the usual application of RL is that our agent maintains a knowledge base of skills that it learns using intrinsic rewards. In most other regards, our extended RL framework is based on putting together learning and planning algorithms for

**Loop forever**

    Current state $s_t$, current primitive action $a_t$, current option $o_t$,
    extrinsic reward $r_t^e$, intrinsic reward $r_t^i$

    Obtain next state $s_{t+1}$

    *//— Deal with special case if next state is salient*
    If $s_{t+1}$ is a salient event *e*
        If option for *e*, $o_e$, does not exist in $O$ (skill-KB)
            Create option $o_e$ in skill-KB;
            Add $s_t$ to $I^{o_e}$ *// initialize initiation set*
            Set $\beta^{o_e}(s_{t+1}) = 1$ *// set termination probability*
        *//— set intrinsic reward value*
        $r_{t+1}^i = \tau[1 - P^{o_e}(s_{t+1}|s_t)]$ *// $\tau$ is a constant multiplier*
    else
        $r_{t+1}^i = 0$

    *//— Update all option models*
    For each option $o \neq o_e$ in skill-KB ($O$)
        If $s_{t+1} \in I^o$, then add $s_t$ to $I^o$ *// grow initiation set*
        If $a_t$ is greedy action for $o$ in state $s_t$
            *//— update option transition probability model*
            $P^o(x|s_t) \overset{\alpha}{\leftarrow} [\gamma(1 - \beta^o(s_{t+1})P^o(x|s_{t+1}) + \gamma\beta^o(s_{t+1})\delta_{s_{t+1}x}]$
            *//— update option reward model*
            $R^o(s_t) \overset{\alpha}{\leftarrow} [r_t^e + \gamma(1 - \beta^o(s_{t+1}))R^o(s_{t+1})]$

    *//— Q-learning update of behavior action-value function*
    $Q_B(s_t, a_t) \overset{\alpha}{\leftarrow} [r_t^e + r_t^i + \gamma \max_{a \in A \cup O} Q_B(s_{t+1}, a)]$

    *//— SMDP-planning update of behavior action-value function*
    For each option o in skill-KB
        $Q_B(s_t, o) \overset{\alpha}{\leftarrow} [R^o(s_t) + \sum_{x \in S} P^o(x|s_t) \max_{a \in A \cup O} Q_B(x, a)]$

    *//— Update option action-value functions*
    For each option $o \in O$ such that $s_t \in I^o$
        $Q^o(s_t, a_t) \overset{\alpha}{\leftarrow} [r_t^e + \gamma \; (\beta^o(s_{t+1}) \times \text{terminal value for option o})$
                              $+\gamma(1 - \beta^o(s_{t+1})) \times \max_{a \in A \cup O} Q^o(s_{t+1}, a)]$
        For each option $o' \in O$ such that $s_t \in I^{o'}$ and $o \neq o'$
        $Q^o(s_t, o') \overset{\alpha}{\leftarrow} R^{o'}(s_t) + \sum_{x \in S} P^{o'}(x|s_t)[\beta^o(x) \times \text{terminal val for option o}$
                              $+((1 - \beta^o(x)) \times \max_{a \in A \cup O} Q^o(x, a))]$

    Choose $a_{t+1}$ using $\epsilon$-greedy policy w.r.to $Q_B$ *// — Choose next action*
    *//— Determine next extrinsic reward*
    Set $r_{t+1}^e$ to the extrinsic reward for transition $s_t, a_t \rightarrow s_{t+1}$

    Set $s_t \leftarrow s_{t+1}$; $a_t \leftarrow a_{t+1}$; $r_t^e \leftarrow r_{t+1}^e$; $r_t^i \leftarrow r_{t+1}^i$

Figure 2: Learning Algorithm. Extrinsic reward is denoted $r^e$ while intrinsic reward is denoted $r^i$. Equations of the form $x \overset{\alpha}{\leftarrow} [y]$ are short for $x \leftarrow (1-\alpha)x + \alpha[y]$. The behavior action value function $Q_B$ is updated using a combination of Q-learning and SMDP planning. Throughout $\gamma$ is a discount factor and $\alpha$ is the step-size. The option action value functions $Q^o$ are updated using intra-option Q-learning. Note that the intrinsic reward is only used in updating $Q_B$ and not any of the $Q^o$.

options [12].

**Behavior** The agent behaves in its environment according to an $\epsilon$-greedy policy with respect to an action-value function $Q_B$ that is learned using a mix of Q-learning and SMDP planning as described in Fig. 2. Initially only the primitive actions are available to the agent. Over time, skills represented internally as options and their models also become available to the agent as action choices. Thus, $Q_B$ maps states $s$ and actions $a$ (both primitive and options) to the expected long-term utility of taking that action $a$ in state $s$.

**Salient Events** In our current implementation we assume that the agent has intrinsic or hardwired notions of interesting or "salient" events in its environment. For example, in the playroom environment we present shortly, the agent finds changes in light and sound intensity to be salient. These are intended to be independent of any specific task and likely to be applicable to many environments.

**Reward** In addition to the usual extrinsic rewards there are occasional intrinsic rewards generated by the agent's critic (see Fig. 1B). In this implementation, the agent's intrinsic reward is generated in a way suggested by the novelty response of dopamine neurons. The intrinsic reward for each salient event is proportional to the error in the prediction of the salient event according to the learned option model for that event (see Fig. 2 for detail).

**Skill-KB** The agent maintains a knowledge base of skills that it has learned in its environment. Initially this may be empty. The first time a salient event occurs, say light turned on, structures to learn an option that achieves that salient event (turn-light-on option) are created in the skill-KB. In addition, structures to learn an option model are also created. So for option $o$, $Q^o$ maps states $s$ and actions $a$ (again, both primitive and options) to the long-term utility of taking action $a$ in state $s$. The option for a salient event terminates with probability one in any state that achieves that event and never terminates in any other state. The initiation set, $I^o$, for an option $o$ is incrementally expanded to includes states that lead to states in the current initiation set.

**Learning** The details of the learning algorithm are presented in Fig. 2.

## 4 Playroom Domain: Empirical Results

We implemented intrinsically motivated RL (of Fig. 2) in a simple artificial "playroom" domain shown in Fig. 3A. In the playroom are a number of objects: a light switch, a ball, a bell, two movable blocks that are also buttons for turning music on and off, as well as a toy monkey that can make sounds. The agent has an eye, a hand, and a visual marker (seen as a cross hair in the figure). The agent's sensors tell it what objects (if any) are under the eye, hand and marker. At any time step, the agent has the following actions available to it: 1) move eye to hand, 2) move eye to marker, 3) move eye one step north, south, east or west, 4) move eye to random object, 5) move hand to eye, and 6) move marker to eye. In addition, if both the eye and and hand are on some object, then natural operations suggested by the object become available, e.g., if both the hand and the eye are on the light switch, then the action of flicking the light switch becomes available, and if both the hand and eye are on the ball, then the action of kicking the ball becomes available (which when pushed, moves in a straight line to the marker).

The objects in the playroom all have potentially interesting characteristics. The bell rings once and moves to a random adjacent square if the ball is kicked into it. The light switch controls the lighting in the room. The colors of any of the blocks in the room are only visible if the light is on, otherwise they appear similarly gray. The blue block if pressed turns music on, while the red block if pressed turns music off. Either block can be pushed and as a result moves to a random adjacent square. The toy monkey makes frightened sounds if simultaneously the room is dark and the music is on and the bell is rung. These objects were designed to have varying degrees of difficulty to engage. For example, to get the monkey to cry out requires the agent to do the following sequence of actions: 1) get its eye to the light switch, 2) move hand to eye, 3) push the light switch to turn the light on, 4) find the blue block with its eye, 5) move the hand to the eye, 6) press the blue block to turn

music on, 7) find the light switch with its eye, 8) move hand to eye, 9) press light switch to turn light off, 10) find the bell with its eye, 11) move the marker to the eye, 12) find the ball with its eye, 13) move its hand to the ball, and 14) kick the ball to make the bell ring. Notice that if the agent has already learned how to turn the light on and off, how to turn music on, and how to make the bell ring, then those learned skills would be of obvious use in simplifying this process of engaging the toy monkey.

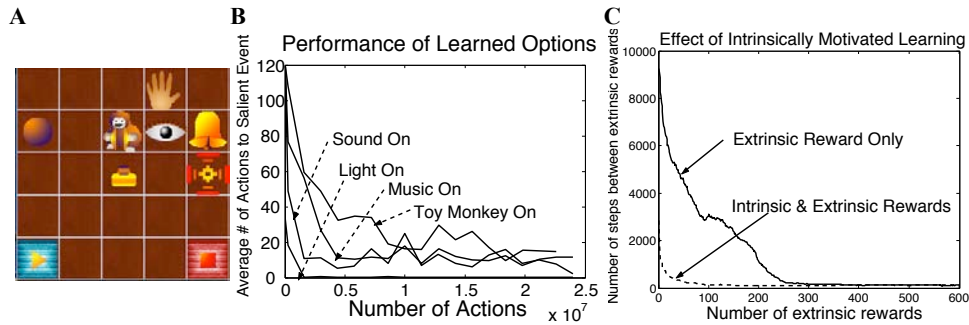

Figure 3: **A.** *Playroom domain.* **B.** *Speed of learning of various skills.* **C.** *The effect of intrinsically motivated learning when extrinsic reward is present. See text for details*

For this simple example, changes in light and sound intensity are considered salient by the playroom agent. Because the initial action value function, $Q_B$, is uninformative, the agent starts by exploring its environment randomly. Each first encounter with a salient event initiates the learning of an option and an option model for that salient event. For example, the first time the agent happens to turn the light on, it initiates the data structures necessary for learning and storing the light-on option. As the agent moves around the environment, all the options (initiated so far) and their models are simultaneously updated using intra-option learning.

As shown in Fig. 2, the intrinsic reward is used to update $Q_B$. As a result, when the agent encounters an unpredicted salient event a few times, its updated action value function drives it to repeatedly attempt to achieve that salient event. There are two interesting side effects of this: 1) as the agent tries to repeatedly achieve the salient event, learning improves both its policy for doing so and its option-model that predicts the salient event, and 2) as its option policy and option model improve, the intrinsic reward diminishes and the agent gets "bored" with the associated salient event and moves on. Of course, the option policy and model become accurate in states the agent encounters frequently. Occasionally, the agent encounters the salient event in a state (set of sensor readings) that it has not encountered before, and it generates intrinsic reward again (it is "surprised").

A summary of results is presented in Fig. 4. Each panel of the figure is for a distinct salient event. The graph in each panel shows both the time steps at which the event occurs as well as the intrinsic reward associated by the agent to each occurrence. Each occurrence is denoted by a vertical bar whose height denotes the amount of associated intrinsic reward. Note that as one goes from top to bottom in this figure, the salient events become harder to achieve and, in fact, become more hierarchical. Indeed, the lowest one for turning on the monkey noise (Non) needs light on, music on, light off, sound on in sequence. A number of interesting results can be observed in this figure. First note that the salient events that are simpler to achieve occur earlier in time. For example, Lon (light turning on) and Loff (light turning off) are the simplest salient events, and the agent makes these happen quite early. The agent tries them a large number of times before getting bored and moving on to other salient events. The reward obtained for each of these events diminishes after repeated exposure to the event. Thus, automatically, the skill of achieving the simpler events are learned before those for the more complex events.

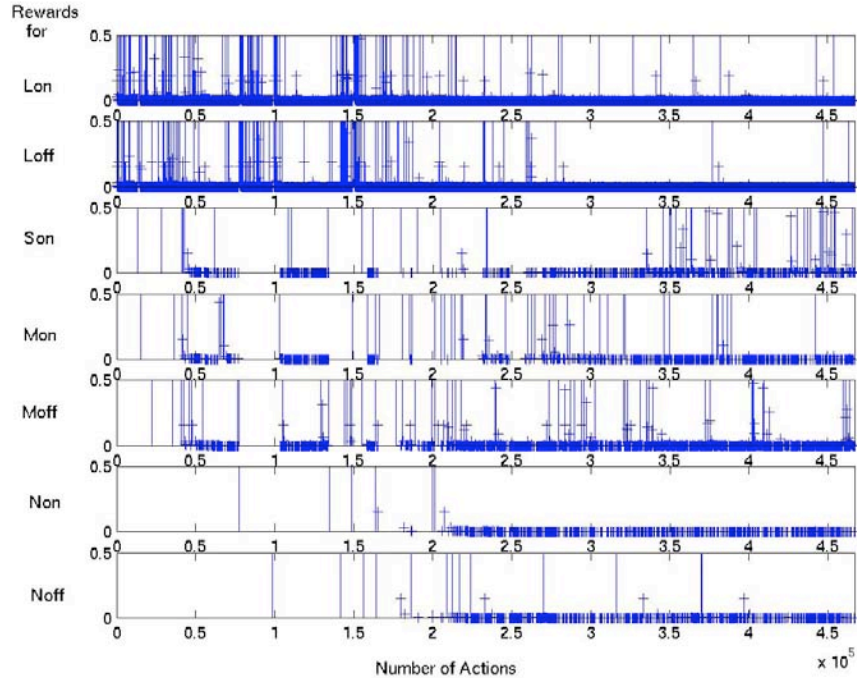

Figure 4: *Results from the playroom domain. Each panel depicts the occurrences of salient events as well as the associated intrinsic rewards. See text for details.*

Of course, the events keep happening despite their diminished capacity to reward because they are needed to achieve the more complex events. Consequently, the agent continues to turn the light on and off even after it has learned this skill because this is a step along the way toward turning on the music, as well as along the way toward turning on the monkey noise. Finally note that the more complex skills are learned relatively quickly once the required sub-skills are in place, as one can see by the few rewards the agent receives for them. The agent is able to bootstrap and build upon the options it has already learned for the simpler events. We confirmed the hierarchical nature of the learned options by inspecting the greedy policies for the more complex options like Non and Noff. The fact that all the options are successfully learned is also seen in Fig. 3B in which we show how long it takes to bring about the events at different points in the agent's experience (there is an upper cutoff of 120 steps). This figure also shows that the simpler skills are learned earlier than the more complex ones.

An agent having a collection of skills learned through intrinsic reward can learn a wide variety of extrinsically rewarded tasks more easily than an agent lacking these skills. To illustrate, we looked at a playroom task in which extrinsic reward was available only if the agent succeeded in making the monkey cry out. This requires the 14 steps described above. This is difficult for an agent to learn if only the extrinsic reward is available, but much easier if the agent can use intrinsic reward to learn a collection of skills, some of which are relevant to the overall task. Fig. 3C compares the performance of two agents in this task. Each starts out with no knowledge of task, but one employs the intrinsic reward mechanism we have discussed above. The extrinsic reward is always available, but only when the monkey cries out. The figure, which shows the average of 100 repetitions of the experiment, clearly shows the advantage of learning with intrinsic reward.

**Discussion**   One of the key aspects of the Playroom example was that intrinsic reward was generated only by unexpected salient events. But this is only one of the simplest

possibilities and has many limitations. It cannot account for what makes many forms of exploration and manipulation "interesting." In the future, we intend to implement computational analogs of other forms of intrinsic motivation as suggested in the psychological, statistical, and neuroscience literatures.

Despite the "toy" nature of this domain, these results are among the most sophisticated we have seen involving intrinsically motivated learning. Moreover, they were achieved quite directly by combining a collection of existing RL algorithms for learning options and option-models with a simple notion of intrinsic reward. The idea of intrinsic motivation for artificial agents is certainly not new, but we hope to have shown that the elaboration of the formal RL framework in the direction we have pursued, together with the use of recently-developed hierarchical RL algorithms, provides a fruitful basis for developing competently autonomous agents.

**Acknowledgement** Satinder Singh and Nuttapong Chentanez were funded by NSF grant CCF 0432027 and by a grant from DARPA's IPTO program. Andrew Barto was funded by NSF grant CCF 0432143 and by a grant from DARPA's IPTO program.

# References

[1] A. G. Barto, S. Singh, and N. Chentanez. Intrinsically motivated learning of hierarchical collections of skills. In *Proceedings of the 3rd International Conference on Developmental Learning (ICDL '04)*, LaJolla CA, 2004.

[2] P. Dayan and B. W. Balleine. Reward, motivation and reinforcement learning. *Neuron*, 36:285–298, 2002.

[3] S. Kakade and P. Dayan. Dopamine: Generalization and bonuses. *Neural Networks*, 15:549–559, 2002.

[4] F. Kaplan and P.-Y. Oudeyer. Motivational principles for visual know-how development. In C. G. Prince, L. Berthouze, H. Kozima, D. Bullock, G. Stojanov, and C. Balkenius, editors, *Proceedings of the Third International Workshop on Epigenetic Robotics : Modeling Cognitive Development in Robotic Systems*, pages 73–80, Edinburgh, Scotland, 2003. Lund University Cognitive Studies.

[5] A. McGovern. *Autonomous Discovery of Temporal Abstractions from Interaction with An Environment.* PhD thesis, University of Massachusetts, 2002.

[6] A. Ng, D. Harada, and S. Russell. Policy invariance under reward transformations: Theory and application to reward shaping. In *Proceedings of the Sixteenth ICML*. Morgan Kaufmann, 1999.

[7] P. Reed, C. Mitchell, and T. Nokes. Intrinsic reinforcing properties of putatively neutral stimuli in an instrumental two-lever discrimination task. *Animal Learning and Behavior*, 24:38–45, 1996.

[8] J. Schmidhuber. A possibility for implementing curiosity and boredom in model-building neural controllers. In *From Animals to Animats: Proceedings of the First International Conference on Simulation of Adaptive Behavior*, pages 222–227, Cambridge, MA, 1991. MIT Press.

[9] W. Schultz. Predictive reward signal of dopamine neurons. *Journal of Neurophysiology*, 80:1–27, 1998.

[10] R. S. Sutton. Integrated modeling and control based on reinforcement learning and dynamic programming. In *Proceedings of NIPS*, pages 471–478, San Mateo, CA, 1991.

[11] R. S. Sutton and A. G. Barto. *Reinforcement Learning: An Introduction*. MIT Press, Cambridge, MA, 1998.

[12] R. S. Sutton, D. Precup, and S. Singh. Between mdps and semi-mdps: A framework for temporal abstraction in reinforcement learning. *Artificial Intelligence*, 112:181–211, 1999.

[13] J. Wang, J. McClelland, A. Pentland, O. Sporns, I. Stockman, M. Sur, and E. Thelen. Autonomous mental develoopment by robots and animals. *Science*, 291:599–600, 2001.

[14] R. W. White. Motivation reconsidered: The concept of competence. *Psychological Review*, 66:297–333, 1959.
